# *Note on Development of Modularity in Simple Cortical Models*

**Alex Chernjavsky**[1]
Neuroscience Graduate Program
Section of Molecular Neurobiology
Howard Hughes Medical Institute
Yale University

**John Moody**[2]
Yale Computer Science
PO Box 2158 Yale Station
New Haven, CT 06520
Email: moody@cs.yale.edu

## ABSTRACT

The existence of modularity in the organization of nervous systems (e.g. cortical columns and olfactory glomeruli) is well known. We show that localized activity patterns in a layer of cells, *collective excitations,* can induce the formation of modular structures in the anatomical connections via a Hebbian learning mechanism. The networks are spatially homogeneous before learning, but the spontaneous emergence of localized collective excitations and subsequently modularity in the connection patterns breaks translational symmetry. This spontaneous symmetry breaking phenomenon is similar to those which drive pattern formation in reaction-diffusion systems. We have identified requirements on the patterns of lateral connections and on the gains of internal units which are essential for the development of modularity. These essential requirements will most likely remain operative when more complicated (and biologically realistic) models are considered.

[1]Present Address: Molecular and Cellular Physiology, Beckman Center, Stanford University, Stanford, CA 94305.
[2]Please address correspondence to John Moody.

## 1    Modularity in Nervous Systems

Modular organization exists throughout the nervous system on many different spatial scales. On the very small scale, synapses appear to be clustered on dendrites. On the very large scale, the brain as a whole is composed of many anatomically and functionally distinct regions. At intermediate scales, the scales of networks and maps, the brain exhibits columnar structures.

The purpose of this work is to suggest possible mechanisms for the development of modular structures at the intermediate scales of networks and maps. The best known modular structure at this scale is the column.  Many modality- specific variations of columnar organization are known, for example orientation selective columns, ocular dominance columns, color sensitive blobs, somatosensory barrels, and olfactory glomeruli. In addition to these anatomically well-established structures, other more speculative modular anatomical structures may exist.  These include the frontal eye fields of association cortex whose modular structure is inferred only from electrophysiology and the hypothetical existence of minicolumns and possibly neuronal groups.

Although a complete biophysical picture of the development of modular structures is still unavailable, it is well established that electrical activity is crucial for the development of certain modular structures such as complex synaptic zones and ocular dominance columns (see Kalil 1989 and references therein). It is also generally conjectured that a Hebb-like mechanism is operative in this development. These observations form a basis for our operating hypothesis described below.

## 2    Operating Hypothesis and Modeling Approach

Our hypothesis in this work is that localized activity patterns in a layer of cells induce the development of modular anatomical structure within the layer. We further hypothesize that the emergence of localized activity patterns in a layer is due to the properties of the intrinsic network dynamics and does not necessarily depend upon the system receiving localized patterns of afferent activity.

Our work therefore has two parts.  First, we show that localized patterns of activity on a preferred spatial scale, *collective excitations*, spontaneously emerge in homogeneous networks with appropriate lateral connectivity and cellular response properties when driven with arbitrary stimulus (see Moody 1990). Secondly, we show that these collective excitations induce the formation of modular structures in the connectivity patterns when coupled to a Hebbian learning mechanism.

The emergence of collective excitations at a preferred spatial scale in a homogeneous network breaks translational symmetry and is an example of spontaneous symmetry breaking.  The Hebbian learning freezes the modular structure into the anatomy. The time scale of collective excitations is short, while the Hebbian learning process occurs over a longer time scale. The spontaneous symmetry breaking mechanism is similar to that which drives pattern formation in reaction-diffusion systems (Turing 1952, Meinhardt 1982). Reaction-diffusion models have been applied to pattern for-

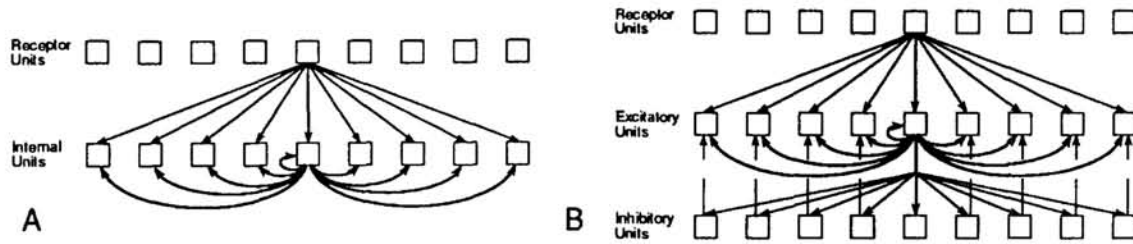

**Figure 1:** Network Models. A: Additive Model. B: Shunting Inhibition Model. Artwork after Pearson et al. (1987).

mation in both biological and physical systems. One of the best known applications is to the development of zebra stripes and leopard spots. Also, a network model with dynamics exhibiting spontaneous symmetry breaking has been proposed by Cowan (1982) to explain geometrical visual hallucination patterns.

Previous work by Pearson et al. (1987) demonstrated empirically that modularity emerged in simulations of an idealized but rather complex model of somatosensory cortex. The Pearson work was purely empirical and did not attempt to analyze theoretically why the modules developed. It provided an impetus, however, for our developing the theoretical results which we present here and in Moody (1990).

Our work is thus intended to provide a possible theoretical foundation for the development of modularity. We have limited our attention to simple models which we can analyze mathematically in order to identify the essential requirements for the formation of modules. To convince ourselves that both collective excitations and the consequent development of modules are somewhat universal, we have considered several different network models. All models exhibit collective excitations. We believe that more biologically realistic (and therefore more complicated) models will very likely exhibit similar behaviors.

This paper is a substantially abbreviated version of Chernjavsky and Moody (1990).

## 3  Network Dynamics: Collective Excitations

The analysis of network dynamics presented in this section is adapted from Moody (1990). Due to space limitations, we present here a detailed analysis of only the simplest model which exhibits collective excitations.

All network models which we consider possess a single layer of receptor cells which provide input to a single internal layer of laterally-connected cells. Two general classes of models are considered (see figure 1): additive models and shunting inhibition models. The additive models contain a single population of internal cells which make both lateral excitatory and inhibitory connections. Both connection types are additive. The shunting inhibition models have two populations of cells in the internal layer: excitatory cells which make additive synaptic axonal contact with other cells and inhibitory cells which shunt the activities of excitatory cells.

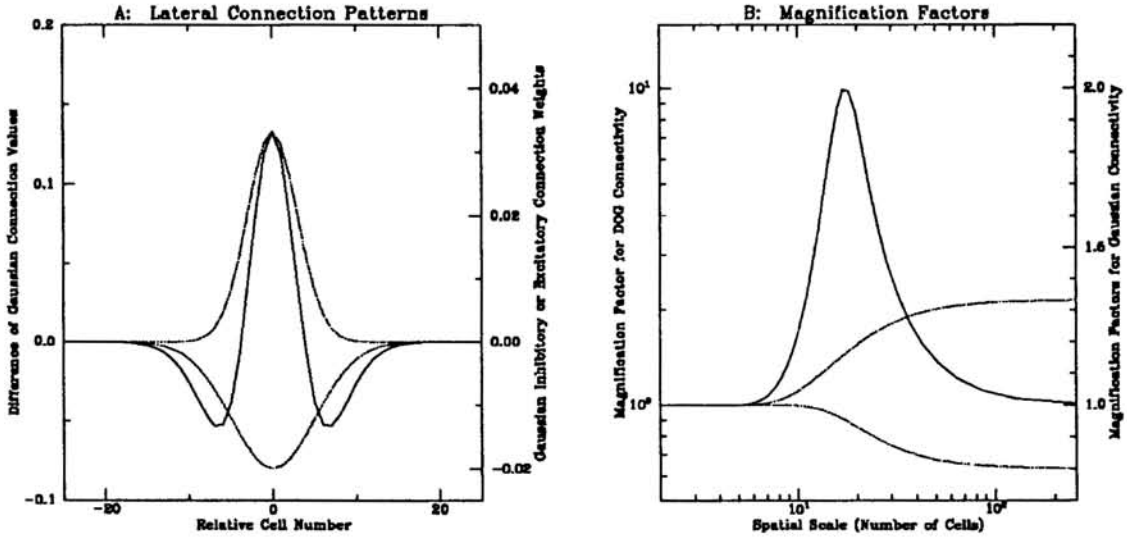

**Figure 2:** A: Excitatory, Inhibitory, and Difference of Gaussian Lateral Connection Patterns. B: Magnification Functions for the Linear Additive Model.

The additive models are further subdivided into models with linear internal units and models with nonlinear (particularly sigmoidal) internal units. The shunting inhibition models have linear excitatory units and sigmoidal inhibitory units. We have considered two variants of the shunting models, those with and without lateral excitatory connections.

For simplicity and tractability, we have limited the use of nonlinear response functions to at most one cell population in all models. More elaborate network models could make greater use of nonlinearity, a greater variety of cell types (eg. disinhibitory cells), and use more ornate connectivity patterns. However, such additional structure can only add richness to the network behavior and is not likely to remove the collective excitation phenomenon.

### 3.1    Dynamics for the Linear Additive Model

To elucidate the fundamental requirements for the spontaneous emergence of collective excitations, we now focus on the minimal model which exhibits the phenomenon, the linear additive model. This model is exactly solvable.

As we will see, collective excitations will emerge provided that the appropriate lateral connectivity patterns are present and that the gains of the internal units are sufficiently high. These basic requirements will carry over to the nonlinear additive and shunting models.

The network relaxation equations for the linear additive model are:

$$\tau_d \frac{d}{dt} V_i = -V_i + \sum_j W_{ij}^{aff} R_j + \sum_j W_{ij}^{lat} E_j \tag{1}$$

where $R_j$ and $E_j$ are the activities (firing rates) of the $j^{th}$ receptor and internal

cells respectively, $V_i$ is the somatic potential of the $i^{th}$ internal cell, $W_{ij}^{aff}$ and $W_{ij}^{lat}$ are the afferent and lateral connections respectively, and $\tau_d$ is the dynamical relaxation time. The somatic potentials and firing rates of the internal units are linearly related by $E_i = (V_i - \theta)/\epsilon$ where $\theta$ is an offset or threshold and $\epsilon^{-1}$ is the gain.

The steady state solutions of the network equations can be solved exactly by reformulating the problem in the continuum limit ($i \mapsto x$):

$$\tau_d \frac{d}{dt} V(x) = -V(x) + A(x) + \int dy \, W^{lat}(x - y) E(y) \qquad (2)$$

$$A(x) \equiv \int dy \, W^{aff}(x - y) R(y) \qquad (3)$$

The functions $R(y)$ and $E(y)$ are activation densities in the receptor and internal layers respectively. $A(x)$ is the integrated input activation density to the internal layer. The functions $W^{aff}(x - y)$ and $W^{lat}(x - y)$ are interpreted as connection densities. Note that the network is spatially homogeneous since the connection densities depend only on the relative separation of post-synaptic and pre-synaptic cells ($x - y$). Examples of lateral connectivity patterns $W^{lat}(x - y)$ are shown in figure 2A. These include local gaussian excitation, intermediate range gaussian inhibition, and a scaled difference of gaussians (DOG).

The exact stationary solution $\frac{d}{dt} V(x) = 0$ of the continuum dynamics of equation 2 can be computed by fourier transforming the equations to the spatial frequency domain. The solution thereby obtained (for $\theta = 0$) is $E(k) = M(k)A(k)$, where the variable $k$ is the spatial frequency and $M(k)$ is the network *magnification function*:

$$M(k) \equiv \frac{1}{\epsilon - W^{lat}(k)}. \qquad (4)$$

Positive magnification factors correspond to stable modes. When the magnification function is large and positive, the network magnifies afferent activity structure on specific spatial scales. This occurs when the inverse gain $\epsilon$ is sufficiently small and/or the fourier transform of the pattern of lateral connectivity $W^{lat}(k)$ has a peak at a non-zero frequency.

Figure 2B shows magnification functions (plotted as a function of spatial scale $2\pi/k$) corresponding to the lateral connectivity patterns shown in figure 2A for a network with $\epsilon = 1$. Note that the gaussian excitatory and gaussian inhibitory connection patterns (which have total integrated weight $\pm 0.25$) magnify structure at large spatial scales by factors of 1.33 and 0.80 respectively. The scale DOG connectivity pattern (which has total weight 0) gives rise to no large scale or small scale magnification, but rather magnifies structure on an intermediate spatial scale of 17 cells.

We illustrate the response of linear networks with unit gain $\epsilon = 1$ and different lateral connectivity patterns in figure 3. The networks correspond to connectivities

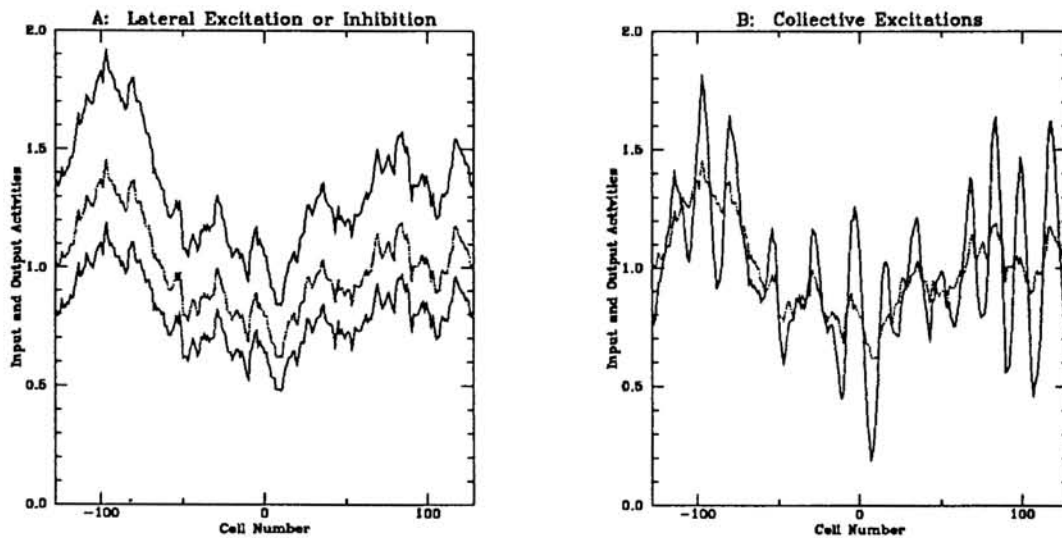

**Figure 3:** Response of a Linear Network to Random Input. A: Response of neutral (dashed), lateral excitatory (upper solid), and lateral inhibitory (lower solid) networks. B: Collective excitations (solid) as response to random input (dashed) in network with DOG lateral connectivity.

and magnification functions shown in figure 2. Part A, shows the response $E(x)$ of neutral, gaussian excitatory, and gaussian inhibitory networks to net afferent input $A(x)$ generated from a random $1/f^2$ noise distribution. The neutral network (no lateral connections) yields the identity response to random input; the networks with the excitatory and inhibitory lateral connection patterns exhibit boosted and reduced response respectively. Part B shows the emergence of collective excitations (solid) for the scaled DOG lateral connectivity. The resulting collective excitations have a typical period of about 17 cells, corresponding to the peak in the magnification function shown in figure 2. Note that the positions of peaks and troughs of the collective excitations correspond approximately to local extrema in the random input (dashed).

It is interesting to note that although the individual components of the networks are all linear, the overall response of the interacting system is nonlinear. It is this collective nonlinearity of the system which enables the emergence of collective excitations. Thus, although the connectivity giving rise to the response in figure 3B is a scaled sum of the connectivities of the excitatory and inhibitory networks of figure 3A, the responses themselves do not add.

## 3.2  Dynamics for Nonlinear Models

The nonlinear models, including the sigmoidal additive model and the shunting models, exhibit the collective excitation phenomenon as well. These models can not be solved exactly, however. See Moody (1990) for a detailed description.

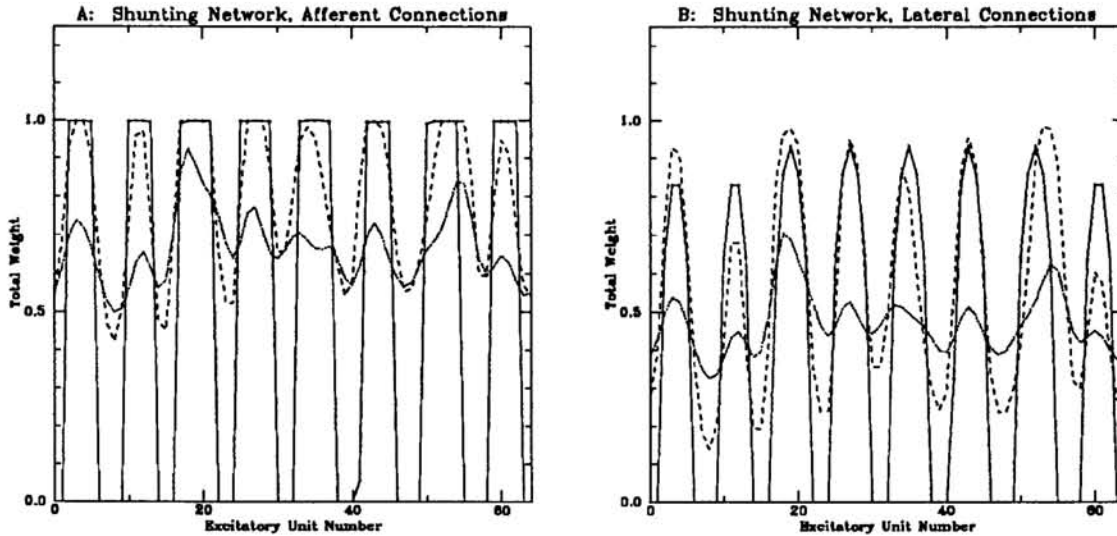

**Figure 4:** Development of Modularity in the Nonlinear Shunting Inhibition Model. Curves represent the average incoming connection value (either afferent connections or lateral connections) for each excitatory internal unit. A: Time development of Afferent Modularity. B: Time development of Lateral Modularity. A and B: 400 iterations (dotted line), 650 iterations (dashed line), 4100 iterations (solid line).

## 4  Hebbian Learning: Development of Modularity

The presence of collective excitations in the network dynamics enables the development of modular structures in both the afferent and lateral connection patterns via Hebbian learning. Due to space limitations, we present simulation results only for the nonlinear shunting model. We focus on this model since it has both afferent and lateral plastic connections and thus develops both afferent and lateral modular connectivities. The other models do not have plastic lateral connections and develop only afferent connectivity modules. A more detailed account of all simulations is given in Chernjavsky and Moody (1990).

In our networks, the plastic excitatory connection values are restricted to the range $W \in [0, 1]$. The homogeneous initial conditions for all connection values are $W = 0.5$. We have considered several variants of Hebbian learning. For the simulations we report here, however, we use only the simple Hebb rule with decay:

$$\tau_{Hebb}\frac{d}{dt}W_{ij} = M_i N_j - \beta \tag{5}$$

where $M_i$ and $N_j$ are the post- and pre-synaptic activities respectively and $\beta$ is the decay constant chosen to be approximately equal to the expected value $\bar{M}\bar{N}$ averaged over the whole network. This choice of $\beta$ makes the Hebb similar to the covariance type rule of Sejnowski (1977). $\tau_{Hebb}$ is the timescale for learning.

The simulation results illustrated in figure 4 are of one dimensional networks with 64 units per layer. In these simulations, the units and connections illustrated are

intended to represent a continuum. The connection densities for afferent and lateral excitatory connections were chosen to be gaussian with a maximum fan-out of 9 lattice units. The inhibitory connection density had a maximum fan-in of 19 lattice units and had a symmetric bimodal shape. The sigmas of the excitatory and inhibitory fan-ins were respectively 1.4 and 2.1 (short-range excitation and longer range inhibition). The linear excitatory units had $\epsilon = 1$ and $\theta = 0$, while the sigmoidal inhibitory units had $\epsilon = 0.125$ and $\theta = 0.5$.

The input activations were uniform random values in the range $[0, 1]$. The input activations were spatially and temporally uncorrelated. Each input pattern was presented for only one dynamical relaxation time of the network (10 timesteps).

The following adaptation rate parameters were used: dynamical relaxation rate $\tau_d^{-1} = 0.1$, learning rate $\tau_{Hebb}^{-1} = 0.01$, weight decay constant $\beta = 0.125$.

## Acknowledgements

The authors wish to thank George Carman, Martha Constantine-Paton, Kamil Grajski, Daniel Kammen, John Pearson, and Gordon Shepherd for helpful comments. A.C. thanks Stephen J Smith for the freedom to pursue projects outside the laboratory. J.M. was supported by ONR Grant N00014-89-J-1228 and AFOSR Grant 89-0478. A.C. was supported by the Howard Hughes Medical Institute and by the Yale Neuroscience Program.

## References

Alex Chernjavsky and John Moody. (1990) Spontaneous development of modularity in simple cortical models. Submitted to *Neural Computation*.

Jack D. Cowan. (1982) Spontaneous symmetry breaking in large scale nervous activity. *Intl. J. Quantum Chemistry*, 22:1059.

Ronald E. Kalil. (1989) Synapse formation in the developing brain. *Scientific American* December.

H. Meinhardt. (1982) *Models of Biological Pattern Formation*. Academic Press, New York.

John Moody. (1990) Dynamics of lateral interaction networks. Technical report, Yale University. (In Preparation.)

Vernon B. Mountcastle. (1957) Modality and topographic properties of single neurons of cat's somatic sensory cortex. *Journal of Neurophysiology*, 20:408.

John C. Pearson, Leif H. Finkel, and Gerald M. Edelman. (1987) Plasticity in the organization of adult cerebral cortical maps: A computer simulation based on neuronal group selection. *Journal of Neuroscience*, 7:4209.

Terry Sejnowski. (1977) Strong covariance with nonlinearly interacting neurons. *J. Math. Biol.* 4:303.

Alan Turing. (1952) The chemical basis of morphogenesis. *Phil. Trans. R. Soc.*, B237:37.